# Parameter Expanded Variational Bayesian Methods

**Yuan (Alan) Qi**
MIT CSAIL
32 Vassar street
Cambridge, MA 02139
alanqi@csail.mit.edu

**Tommi S. Jaakkola**
MIT CSAIL
32 Vassar street
Cambridge, MA 02139
tommi@csail.mit.edu

## Abstract

Bayesian inference has become increasingly important in statistical machine learning. Exact Bayesian calculations are often not feasible in practice, however. A number of approximate Bayesian methods have been proposed to make such calculations practical, among them the variational Bayesian (VB) approach. The VB approach, while useful, can nevertheless suffer from slow convergence to the approximate solution. To address this problem, we propose Parameter-eXpanded Variational Bayesian (PX-VB) methods to speed up VB. The new algorithm is inspired by parameter-expanded expectation maximization (PX-EM) and parameter-expanded data augmentation (PX-DA). Similar to PX-EM and -DA, PX-VB expands a model with auxiliary variables to reduce the coupling between variables in the original model. We analyze the convergence rates of VB and PX-VB and demonstrate the superior convergence rates of PX-VB in variational probit regression and automatic relevance determination.

## 1 Introduction

A number of approximate Bayesian methods have been proposed to offset the high computational cost of exact Bayesian calculations. Variational Bayes (VB) is one popular method of approximation. Given a target probability distribution, variational Bayesian methods approximate the target distribution with a factored distribution. While factoring omits dependencies present in the target distribution, the parameters of the factored approximation can be adjusted to improve the match. Specifically, the approximation is optimized by minimizing the KL-divergence between the factored distribution and the target. This minimization can be often carried out iteratively, one component update at a time, despite the fact that the target distribution may not lend itself to exact Bayesian calculations. Variational Bayesian approximations have been widely used in Bayesian learning (e.g., (Jordan et al., 1998; Beal, 2003; Bishop & Tipping, 2000)).

Variational Bayesian methods nevertheless suffer from slow convergence when the variables in the factored approximation are actually strongly coupled in the original model. The same problem arises in popular Gibbs sampling algorithm. The sampling process converges slowly in cases where the variables are strongly correlated. The slow convergence can be alleviated by data augmentation (van Dyk & Meng, 2001; Liu & Wu, 1999), where the idea is to identify an optimal reparameterization (within a family of possible reparameterizations) so as to remove coupling. Similarly, in a deterministic context, Liu et al. (1998) proposed over-parameterization of the model to speed up EM convergence. Our work here is inspired by DA sampling and PX-EM. Our approach uses auxiliary parameters to speed up the deterministic approximation of the target distribution.

Specifically, we propose Parameter-eXpanded Variational Bayesian (PX-VB) method. The original model is modified by auxiliary parameters that are optimized in conjunction with the variational approximation. The optimization of the auxiliary parameters corresponds to a parameterized joint

optimization of the variational components; the role of the new updates is to precisely remove otherwise strong functional couplings between the components thereby facilitating fast convergence.

## 2 An illustrative example

Consider a toy Bayesian model, which has been considered by Liu and Wu (1999) for sampling.

$$p(y|w, z) = \mathcal{N}(y \mid w + z, 1), \qquad p(z) = \mathcal{N}(z \mid \mathbf{0}, D) \tag{1}$$

where $D$ is a know hyperparameter and $p(w) \propto 1$. The task is to compute the posterior distribution of $w$. Suppose we use a VB method to approximate $p(w|y)$, $p(z|y)$ and $p(w, z|y)$ by $q(w)$, $q(z)$ and $q(w, z) = q(w)q(z)$, respectively. The approximation is optimized by minimizing $KL(q(w)q(z)\|p(y|w, z)p(z))$ (the second argument need not be normalized). The general forms of the component updates are given by

$$q(w) \propto \exp(\langle \ln p(y|w, z)p(z) \rangle_{q(z)}) \tag{2}$$

$$q(z) \propto \exp(\langle \ln p(y|w, z)p(z) \rangle_{q(w)}) \tag{3}$$

It is easy to derive the updates in this case:

$$q(w) = \mathcal{N}(w|y - \langle z \rangle, 1) \qquad q(z) = \mathcal{N}(z|\frac{y - \langle w \rangle}{1 + D^{-1}}, \frac{1}{1 + D^{-1}}) \tag{4}$$

Now let us analyze the convergence of the mean parameter of $q(\mathbf{w})$, $\langle w \rangle = y - \langle z \rangle$. Iteratively,

$$\langle w \rangle = \frac{D^{-1}}{1 + D^{-1}}y + \frac{\langle w \rangle}{1 + D^{-1}} = D^{-1}\big((1 + D^{-1})^{-1}y + (1 + D^{-1})^{-2}y + \cdots \big) = y.$$

The variational estimate $\langle w \rangle$ converges to $y$, which actually is the true posterior mean (For this toy problem, $p(w|y) = \mathcal{N}(w|y, 1+D)$). Furthermore, if $D$ is large, $\langle w \rangle$ converges slowly. Note that the variance parameter of $q(\mathbf{w})$ converges to $1$ in one iteration, though underestimates the true posterior variance $1 + D$.

Intuitively, the convergence speed of $\langle w \rangle$ and $q(w)$ suffers from strong coupling between the updates of $w$ and $z$. In other words, the update information has to go through a feedback loop $w \to z \to w \cdots$. To alleviate the coupling, we expand the original model with an additional parameter $\alpha$:

$$p(y|w, z) = \mathcal{N}(y \mid w + z, 1) \qquad p(z|\alpha) = \mathcal{N}(z \mid \alpha, D) \tag{5}$$

The expanded model reduces to the original one when $\alpha$ equals the null value $\alpha_0 = 0$.

Now having computed $q(z)$ given $\alpha = 0$, we minimize $KL(q(w)q(z)\|p(y|w, z)p(z|\alpha))$ over $\alpha$ and obtain the minimizer $\alpha = \langle z \rangle$. Then, we reduce the expanded model to the original one by applying the reduction rule

$$z^{\text{new}} = z - \alpha = z - \langle z \rangle, \qquad w^{\text{new}} = w + \alpha = w + \langle z \rangle.$$

Correspondingly, we change the measures of $q(w)$ and $q(z)$:

$$q(w + \langle z \rangle) \to q(w^{\text{new}}) = \mathcal{N}(w^{\text{new}}|y, 1) \qquad q(z - \langle z \rangle) \to q(z^{\text{new}}) = \mathcal{N}(z^{\text{new}}|0, \frac{1}{1 + D^{-1}}) \tag{6}$$

Thus, the PX-VB method converges. Here $\alpha$ breaks the update loop between $q(w)$ and $q(z)$ and plays the role of a correction force; it corrects the update trajectories of $q(w)$ and $q(z)$ and makes them point directly to the convergence point.

## 3 The PX-VB Algorithm

In the general PX-VB formulation, we over-parameterize the model $p(\hat{\mathbf{x}}, D)$ to get $p_{\boldsymbol{\alpha}}(\mathbf{x}, D)$, where the original model is recovered for some default values of the auxiliary parameters $\boldsymbol{\alpha} = \boldsymbol{\alpha}_0$. The algorithm consists of the typical VB updates relative to $p_{\boldsymbol{\alpha}}(\mathbf{x}, D)$, the optimization of auxiliary parameters $\boldsymbol{\alpha}$, as well as a reduction step to turn the model back to the original form where $\boldsymbol{\alpha} = \boldsymbol{\alpha}_0$. This last reduction step has the effect of jointly modifying the components of the factored variational approximation. Put another way, we push the change in $p_{\boldsymbol{\alpha}}(\mathbf{x}, D)$, due to the optimization of $\boldsymbol{\alpha}$, into the variational approximation instead. Changing the variational approximation in this manner permits us to return the model into its original form and set $\boldsymbol{\alpha} = \boldsymbol{\alpha}_0$.

Specifically, we first expand $p(\hat{\mathbf{x}}, D)$ to obtain $p_{\boldsymbol{\alpha}}(\mathbf{x}, D)$. Then at the $t^{th}$ iteration,

1. $q(\mathbf{x}_s)$ are updated sequentially. Note that the approximate distribution $q(\mathbf{x}) = \prod_s q(\mathbf{x}_s)$.

2. We minimize $KL(q(\mathbf{x})\|p_{\boldsymbol{\alpha}}(\mathbf{x}, D))$ over the auxiliary parameters $\boldsymbol{\alpha}$. This optimization can be done jointly with some components of the variational distribution, if feasible.

3. The expanded model is reduced to the original model through reparameterization. Accordingly, we change $q^{(t+1)}(\mathbf{x})$ to $q^{(t+1)}(\hat{\mathbf{x}})$ such that

$$KL(q^{(t+1)}(\hat{\mathbf{x}})\|p_{\boldsymbol{\alpha}_0}(\hat{\mathbf{x}}, D)) = KL(q(\mathbf{x})\|p_{\boldsymbol{\alpha}^{(t+1)}}(\mathbf{x}, D)) \tag{7}$$

where $q^{(t+1)}(\hat{\mathbf{x}})$ are the modified components of the variational approximation.

4. Set $\boldsymbol{\alpha} = \boldsymbol{\alpha}_0$.

Since each update of PX-VB decreases or maintains the KL divergence $KL(q(\mathbf{x})\|p(\mathbf{x}, D))$, which is lower bounded, PX-VB reaches a stationary point for $KL(q(\mathbf{x})\|p(\mathbf{x}, D))$. Empirically, PX-VB often achieves solution similar to what VB achieves, with faster convergence.

A simple strategy to implement PX-VB is to use a mapping $S_{\boldsymbol{\alpha}}$, parameterized by $\boldsymbol{\alpha}$, over the variables $\hat{\mathbf{x}}$. After sequentially optimizing over the components $\{q(\mathbf{x}_s)\}$, we maximize $\langle \ln p_{\boldsymbol{\alpha}}(\mathbf{x}) \rangle_{q(\mathbf{x})}$ over $\boldsymbol{\alpha}$. Then, we reduce $p_{\boldsymbol{\alpha}}(\mathbf{x}, D)$ to $p(\hat{\mathbf{x}}, D)$ and $q(\mathbf{x})$ to $q(\hat{\mathbf{x}})$ through the inverse mapping of $S_{\boldsymbol{\alpha}}$, $M_{\boldsymbol{\alpha}} \equiv S_{\boldsymbol{\alpha}}^{-1}$. Since we optimize $\boldsymbol{\alpha}$ after optimizing $\{q(\hat{\mathbf{x}}_s)\}$, the mapping $S$ should change at least two components of $\mathbf{x}$. Otherwise, the optimization over $\boldsymbol{\alpha}$ will do nothing since we have already optimized over each $q(\hat{\mathbf{x}}_s)$. If we jointly optimize $\boldsymbol{\alpha}$ and one component $q(\mathbf{x}_s)$, it suffices (albeit need not be optimal) for the mapping $S_{\boldsymbol{\alpha}}$ to change only $q(\mathbf{x}_s)$.

Algorithmically, PX-VB bears a strong similarity to PX-EM (Liu et al., 1998). They both expand the original model and both are based on lower bounding KL-divergence. However, the key difference is that the reduction step in PX-VB changes the lower-bounding distributions $\{q(\mathbf{x}_s)\}$, while in PX-EM the reduction step is performed only for the parameters in $p(\mathbf{x}, D)$. We also note that the PX-VB reduction step via $M_{\boldsymbol{\alpha}}$ leaves the KL-divergence (lower bound on the likelihood) invariant, while in PX-EM the likelihood of the observed data remains the same after the reduction. Because of these differences, general EM acceleration methods (e.g., (Salakhutdinov et al., 2003)) can not be directly applied to speed up VB convergence.

In the following sections, we present PX-VB methods for two popular Bayesian models: Probit regression for data classification and Automatic Relevance Determination (ARD) for feature selection and sparse learner.

### 3.1 Bayesian Probit regression

Probit regression is a standard classification technique (see, e.g., (Liu et al., 1998) for the maximum likelihood estimation). Here we demonstrate the use of variational Bayesian methods to train Probit models.

The data likelihood for Probit regression is

$$p(\mathbf{t}|\mathbf{X}, \mathbf{w}) = \prod_n \sigma(t_n \mathbf{w}^{\mathrm{T}} \mathbf{x}_n),$$

where $\mathbf{X} = [\mathbf{x}_1, \ldots, \mathbf{x}_N]$ and $\sigma$ is the standard normal cumulative distribution function. We can rewrite the likelihood in an equivalent form

$$p(t_n|z_n) = \mathrm{sign}(t_n z_n) \qquad\qquad p(z_n|\mathbf{w}, \mathbf{x}_n) = \mathcal{N}(z_n|\mathbf{w}^{\mathrm{T}} \mathbf{x}_n, 1) \tag{8}$$

Given a Gaussian prior over the parameter, $p(\mathbf{w}) = \mathcal{N}(\mathbf{w}|0, v_0 \mathbf{I})$, we wish to approximate the posterior distribution $p(\mathbf{w}, \mathbf{z}|\mathbf{X}, \mathbf{t})$ by $q(\mathbf{w}, \mathbf{z}) = q(\mathbf{w}) \prod_n q(z_n)$. Minimizing $KL(q(\mathbf{w}) \prod_n q(z_n)\|p(\mathbf{w}, \mathbf{z}, \mathbf{t}|\mathbf{X}))$, we obtain the following VB updates:

$$q(z_n) = \mathcal{TN}(z_n|\langle \mathbf{w} \rangle^{\mathrm{T}} \mathbf{x}_n, 1, t_n z_n) \tag{9}$$

$$q(\mathbf{w}) = \mathcal{N}(\mathbf{w}|(\mathbf{X}\mathbf{X}^{\mathrm{T}} + v_0^{-1}\mathbf{I})^{-1}\mathbf{X}\langle \mathbf{z} \rangle, (\mathbf{X}\mathbf{X}^{\mathrm{T}} + v_0^{-1}\mathbf{I})^{-1}) \tag{10}$$

where $\mathcal{TN}(z_n|\langle \mathbf{w} \rangle^{\mathrm{T}} \mathbf{x}_n, 1, t_n z_n)$ stands for a truncated Gaussian such that $\mathcal{TN}(z_n|\langle \mathbf{w} \rangle^{\mathrm{T}} \mathbf{x}_n, 1, t_n z_n) = \mathcal{N}(z_n|\langle \mathbf{w} \rangle^{\mathrm{T}} \mathbf{x}_n, 1)$ when $t_n z_n > 0$, and it equals 0 otherwise.

To speed up the convergence of the above iterative updates, we apply the PX-VB method. First, we expand the orginal model $p(\hat{\mathbf{w}}, \hat{\mathbf{z}}, \mathbf{t}|\mathbf{X})$ to $p_c(\mathbf{w}, \mathbf{z}, \mathbf{t}|\mathbf{X})$ with the mapping

$$\mathbf{w} = \hat{\mathbf{w}}c \qquad \mathbf{z} = \hat{\mathbf{z}}c \tag{11}$$

such that

$$p_c(z_n|\mathbf{w}, \mathbf{x}_n) = \mathcal{N}(z_n|w^{\mathrm{T}}\mathbf{x}_n, c^2) \qquad p(\mathbf{w}) = \mathcal{N}(\mathbf{w}|\mathbf{0}, c^2 v_0 \mathbf{I}) \tag{12}$$

Setting $c = c_0 = 1$ in the expanded model, we update $q(z_n)$ and $q(\mathbf{w})$ as before, via (9) and (10). Then, we minimize $KL\big(q(\mathbf{z})q(\mathbf{w})\|p_c(\mathbf{w}, \mathbf{z}, \mathbf{t}|\mathbf{X})\big)$ over $c$, yielding

$$c^2 = \frac{1}{N + M}\Big(\sum_n(\langle z_n^2\rangle - 2\langle z_n\rangle\langle\mathbf{w}\rangle^{\mathrm{T}}\mathbf{x}_n + \mathbf{x}_n^{\mathrm{T}}\langle\mathbf{w}\mathbf{w}^{\mathrm{T}}\rangle\mathbf{x}_n) + v_0^{-1}\langle\mathbf{w}\mathbf{w}^{\mathrm{T}}\rangle\Big) \tag{13}$$

where $M$ is the dimension of $\mathbf{w}$. In the degenerate case where $v_0 = \infty$, the denominator of the above equation becomes $N$ instead of $N + M$. Since this equation can be efficiently calculated, the extra computational cost induced by the auxiliary variable is therefore small. We omit the details.

The transformation back to $p_{c_0}$ can be made via the inverse map

$$\widehat{\mathbf{w}} = \mathbf{w}/c \qquad \widehat{\mathbf{z}} = \mathbf{z}/c. \tag{14}$$

Accordingly, we change $q(\mathbf{w})$ to obtain a new posterior approximation $q_c(\widehat{\mathbf{w}})$:

$$q_c(\widehat{\mathbf{w}}) = \mathcal{N}(\widehat{\mathbf{w}}|(\mathbf{X}\mathbf{X}^{\mathrm{T}} + v_0^{-1}\mathbf{I})^{-1}\mathbf{X}\langle\mathbf{z}\rangle/c, (\mathbf{X}\mathbf{X}^{\mathrm{T}} + v_0^{-1}\mathbf{I})^{-1}/c^2) \tag{15}$$

We do not actually need to compute $q_c(z_n)$ if this component will be optimized next.

By changing variables $\mathbf{w}$ to $\widehat{\mathbf{w}}$ through (14), the KL divergence between the approximate and exact posteriors remains the same. After obtaining new approximations $q_c(\widehat{\mathbf{w}})$ and $q(\hat{z}_n)$, we reset $c = c_0 = 1$ for the next iteration.

Though similar to the PX-EM updates for the Probit regression problem (Liu et al., 1998), the PX-VB updates are geared towards providing an approximate posterior distribution.

We use both synthetic data and a kidney biopsy data (van Dyk & Meng, 2001) as numerical examples for probit regression. We set $v_0 = \infty$ in the experiment. The comparison of convergence speeds for VB and PXVB is illustrated in figure 1.

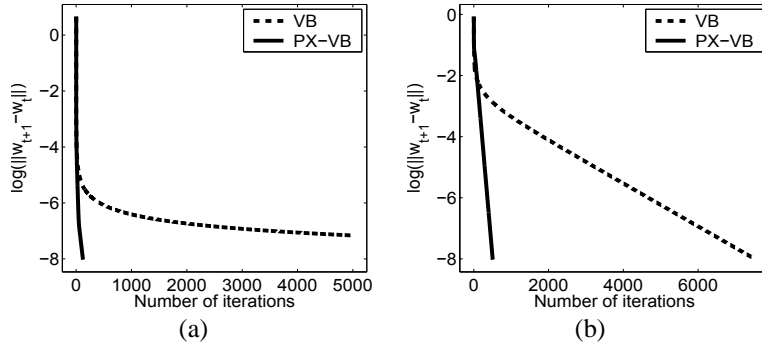

Figure 1: Comparison between VB and PX-VB for probit regression on synthetic (a) and kidney-biospy data sets (b). PX-VB converges significantly faster than VB. Note that the Y axis shows the difference between two consecutive estimates of the posterior mean of the parameter $\mathbf{w}$.

For the synthetic data, we randomly sample a classifier and use it to define the data labels for sampled inputs. We have 100 training and 500 test data points, each of which is 20 features. The kidney data set has 55 data points, each of which is a 3 dimensional vector. On the synthetic data, PX-VB converges immediately while VB updates are slow to converge. Both PX-VB and VB trained classifiers achieve zero test error. On the kidney biopsy data set, PX-VB converges in 507 iterations, while VB converges in 7518 iterations. In other words, PX-VB requires 15 times fewer iterations than VB. In terms of CPU time, which reflects the extra computational cost induced by the auxiliary variables, PX-VB is 14 times more efficient. Among all these runs, PX-VB and VB achieve very similar estimates of the model parameters and the same prediction results. In sum, with a simple modification of VB updates, we significantly improve the convergence speed of variational Bayesian estimation for probit model.

## 3.2 Automatic Relevance Determination

Automatic relevance determination (ARD) is a powerful Bayesian sparse learning technique (MacKay, 1992; Tipping, 2000; Bishop & Tipping, 2000). Here, we focus on variational ARD proposed by Bishop and Tipping (2000) for sparse Bayesian regression and classification.

The likelihood for ARD regression is

$$p(\mathbf{t}|\mathbf{X}, \mathbf{w}, \tau) = \prod_n \mathcal{N}(t_n|\mathbf{w}^{\mathrm{T}}\boldsymbol{\phi}_n, \tau^{-1})$$

where $\boldsymbol{\phi}_n$ is a feature vector based on $\mathbf{x}_n$, such as $[k(\mathbf{x}_1, \mathbf{x}_n), \ldots, [k(\mathbf{x}_N, \mathbf{x}_n)]^{\mathrm{T}}$ where $k(\mathbf{x}_i, \mathbf{x}_j)$ is a nonlinear basis function. For example, we can choose a radial basis function $k(\mathbf{x}_i, \mathbf{x}_j) = \exp(-\|\mathbf{x}_i - \mathbf{x}_j\|/(2\lambda^2)$, where $\lambda$ is the kernel width.

In ARD, we assign a Gaussian prior on the model parameters $\mathbf{w}$: $p(\mathbf{w}|\boldsymbol{\alpha}) = \prod_{m=0}^M \mathcal{N}(w_m|0, \alpha_m^{-1})$, where the inverse variance $\mathrm{diag}(\boldsymbol{\alpha})$ follows a factorized Gamma distribution:

$$p(\boldsymbol{\alpha}) = \prod_m \mathrm{Gamma}(\alpha_m|a, b) = \prod_m b^a \alpha_m^{a-1} e^{-b\alpha_m}/\Gamma(a) \tag{16}$$

where $a$ and $b$ are hyperparameters of the model. The posterior does not have a closed form. Let us approximate $p(\mathbf{w}, \boldsymbol{\alpha}, \tau|\mathbf{X}, \mathbf{t})$ by a factorized distribution $q(\mathbf{w}, \boldsymbol{\alpha}, \tau) = q(\mathbf{w})q(\boldsymbol{\alpha})q(\tau)$. The sequential VB updates on $q(\tau)$, $q(\mathbf{w})$ and $q(\boldsymbol{\alpha})$ are described by Bishop and Tipping (2000).

The variational RVM achieves good generalization performance as demonstrated by Bishop and Tipping (2000). However, its training based on the VB updates can be quite slow. We apply PX-VB to address this issue.

First, we expand the original model $p(\widehat{\mathbf{w}}, \hat{\boldsymbol{\alpha}}, \hat{\tau}|\mathbf{X}, \mathbf{t})$ via

$$\mathbf{w} = \widehat{\mathbf{w}}/r \tag{17}$$

while maintaining $\hat{\boldsymbol{\alpha}}$ and $\hat{\tau}$ unchanged. Consequently, the data likelihood and the prior on $\mathbf{w}$ become

$$p_r(\mathbf{t}|\mathbf{w}, \mathbf{X}, \tau) = \prod_n \mathcal{N}(t_n|r\mathbf{w}^{\mathrm{T}}\boldsymbol{\phi}_n, \tau^{-1}) \qquad p_r(\mathbf{w}|\boldsymbol{\alpha}) = \prod_{m=0}^M \mathcal{N}(w_m|0, r^{-2}\alpha_m^{-1}) \tag{18}$$

Setting $r = r_0 = 1$, we update $q(\tau)$ and $q(\boldsymbol{\alpha})$ as in the regular VB. Then, we want to joint optimize over $q(\mathbf{w})$ and $r$. Instead of performing a fully joint optimization, we optimize $q(\mathbf{w})$ and $r$ separately at the same time. This gives

$$r = \frac{g + \sqrt{g^2 + 16Mf}}{4f} \tag{19}$$

where $f = \langle \tau \rangle \sum_n \mathbf{x}_n^{\mathrm{T}}\langle \mathbf{w}\mathbf{w}^{\mathrm{T}}\rangle \mathbf{x}_n + \sum_m \langle \mathbf{w}_m^2 \rangle \langle \alpha_m \rangle$ and $g = 2\langle \tau \rangle \sum_m \langle \mathbf{w}^{\mathrm{T}}\rangle \mathbf{x}_n t_n$. where $\langle \mathbf{w}^{\mathrm{T}}\rangle$ and $\langle \mathbf{w}\mathbf{w}^{\mathrm{T}}\rangle$ are the first and second order moments of the previous $q(\mathbf{w})$. Since both $f$ and $\mathbf{X}^{\mathrm{T}}\langle \mathbf{w}\rangle$ has been computed previously in VB updates, the added computational cost for $r$ is negligible overall. The separate optimization over $q(\mathbf{w})$ and $r$ often decreases the KL divergence. But it cannot guarantee to achieve a smaller KL divergence than what optimization only over $q(\mathbf{w})$ would achieves. If the regular update over $q(\mathbf{w})$ achieves a smaller KL divergence, we reset $r = 1$.

Given $r$ and $q(\mathbf{w})$, we use $\widehat{\mathbf{w}} = r\mathbf{w}$ to reduce the expanded model to the original one. Correspondingly, we change $q(\mathbf{w}) = \mathcal{N}(\mathbf{w}|\boldsymbol{\mu}_w, \Sigma_w)$ via this reduction rule to obtain $q_r(\hat{\mathbf{w}}) = \mathcal{N}(\widehat{\mathbf{w}}|r\boldsymbol{\mu}_w, r^2\Sigma_w)$.

We can also introduce another auxiliary variable $s$ such that $\boldsymbol{\alpha} = \hat{\boldsymbol{\alpha}}/s$. Similar to the above procedure, we optimize over $s$ the expected log joint probability of the expanded model, and at the same time update $q(\boldsymbol{\alpha})$. Then we change $q(\boldsymbol{\alpha})$ back to $q_s(\hat{\boldsymbol{\alpha}})$ using the inverse mapping $\hat{\boldsymbol{\alpha}} = s\boldsymbol{\alpha}$. Due to the space limitation, we skip the details here.

The auxiliary variables $r$ and $s$ change the individual approximate posteriors $q(\mathbf{w})$ and $q(\boldsymbol{\alpha})$ separately. We can combine these two variables into one and use it to adjust $q(\mathbf{w})$ and $q(\boldsymbol{\alpha})$ jointly. Specifically, we introduce the variable $c$:

$$\mathbf{w} = \widehat{\mathbf{w}}/c \qquad \boldsymbol{\alpha} = c^2\widehat{\boldsymbol{\alpha}}.$$

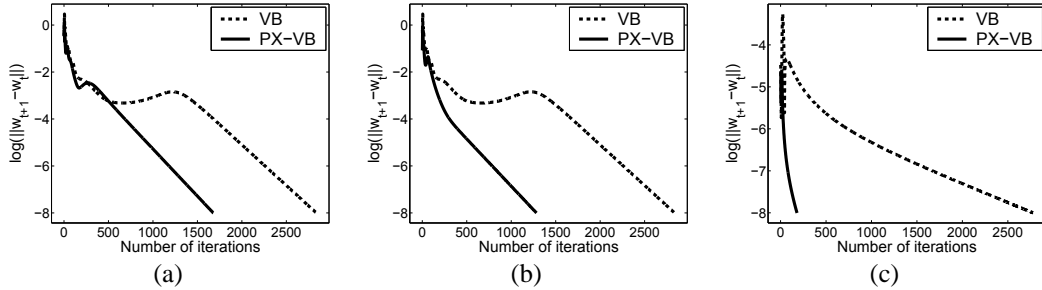

Figure 2: Convergence comparison between VB and PX-VB for ARD regression on synthetic data (a,b) and gene expression data (c). The PX-VB results in (a) and (c) are based on independent auxiliar variables on $\mathbf{w}$ and $\boldsymbol{\alpha}$. The PX-VB result in (b) is based on the auxiliar variable that correlates both $\mathbf{w}$ and $\boldsymbol{\alpha}$. The added computational cost for PX-VB in each iteraction is negligible overall.

Setting $c = c_0 = 1$, we perform the regular updates over $q(\tau)$, $q(\mathbf{w})$ and $q(\boldsymbol{\alpha})$. Then we optimize over $c$ the expected log joint probablity of the expanded model. We cannot find a closed-form solution for the maximization. But we can efficiently compute its gradient and Hessian. Therefore, we perform a few steps of Newton updates to partially optimize $c$. Again, the additional computational cost for calculating $c$ is small. Then using the inverse mapping, we reduce the expanded model to the original one and adjust both $q(\mathbf{w})$ and $q(\boldsymbol{\alpha})$ accordingly. Empirically, this approach can achieve faster convergence than using auxiliary variables on $q(\mathbf{w})$ and $q(\boldsymbol{\alpha})$ separately. This is demonstrated in figure 2(a) and (b).

We compare the convergence speed of VB and PX-VB for the ARD model on both synthetic data and gene expression data. The synthetic data are sampled from the function $\text{sinc}(x) = (\sin x)/x$ for $x \in (-10, 10)$ with added Gaussian noise. We use RBF kernels for the feature expansion $\boldsymbol{\phi}_n$ with kernel width 3. VB and PX-VB provide basically identical predictions. For gene expression data, we apply ARD to analyze the relationship between binding motifs and the expression of their target genes. For this task, we use 3 order polynomial kernels.

The results of convergence comparison are shown in figure 2. With a little modification of VB updates, we increase the convergence speed significantly. Though we demonstrate PX-VB improvement only for ARD regression, the same technique can be used to speed up ARD classification.

## 4 Convergence properties of VB and PX-VB

In this section, we analyze convergence of VB and PX-VB, and their convergence rates.

Define the mapping $\mathbf{q}^{(t+1)} = M(\mathbf{q}^{(t)})$ as one VB update of all the approximate distributions.

Define an objective function as the unnormalized KL divergence:

$$Q(\mathbf{q}) = \int \prod q_i(\mathbf{x}) \log \frac{\prod q_i(\mathbf{x})}{p(\mathbf{x})}) + (\int p(\mathbf{x})\mathrm{d}\mathbf{x} - \int \prod q_i(\mathbf{x})\mathrm{d}\mathbf{x}). \tag{20}$$

It is easy to check that minimizing $Q(\mathbf{q})$ gives the same updates as VB which minimizes KL divergence.

Based on Theorem 2.1 by Luo and Tseng (1992), an iterative application of this mapping to minimize $Q(\mathbf{q})$ results in at least linear convergence to an element $\mathbf{q}^\star$ in the solution set.

Define the mapping $\mathbf{q}^{(t+1)} = M_x(\mathbf{q}^{(t)})$ as one PX-VB update of all the approximate distributions. The convergence of PX-VB follows from similar arguments. i.e., $\boldsymbol{\beta} = [\mathbf{q}^\mathrm{T} \boldsymbol{\alpha}^\mathrm{T}]^\mathrm{T}$ converges to $[\mathbf{q}^{\star\mathrm{T}} \boldsymbol{\alpha}_0^\mathrm{T}]^\mathrm{T}$, where $\boldsymbol{\alpha} \in \Lambda$ are the expanded model parameters, $\boldsymbol{\alpha}_0$ are the null value in the original model.

### 4.1 Convergence rate of VB and PX-VB

The matrix rate of convergence $DM(\mathbf{q})$:

$$\mathbf{q}^{(t+1)} - \mathbf{q}^\star = DM(\mathbf{q})^\mathrm{T}(\mathbf{q}^{(t)} - \mathbf{q}^\star) \tag{21}$$

where $DM(\mathbf{q}) = \left(\frac{\partial M_j(\mathbf{q})}{\partial q_i}\right)$.

Define the global rate of convergence for $\mathbf{q}$: $r = \lim_{t\to\infty} \frac{\|\mathbf{q}^{(t+1)} - \mathbf{q}^\star\|}{\|\mathbf{q}^{(t)} - \mathbf{q}^\star\|}$. Under certain regularity conditions, $r = $ the largest eigenvalue of $DM(\mathbf{q})$. The smaller $r$ is, the faster the algorithm converges.

Define the constraint set $\mathbf{g}_s$ as the constraints for the $s^{th}$ update. Then the following theorem holds:

**Theorem 4.1** *The matrix convergence rate for VB is:*

$$DM(\mathbf{q}^\star) = \prod_{s=1}^{S} P_s \tag{22}$$

*where $P_s = \mathbf{B}_s[\mathbf{B}_s^{\mathrm{T}}\left(D^2 Q(\mathbf{q}^\star)\right)^{-1}\mathbf{B}_s]^{-1}\mathbf{B}_s^{\mathrm{T}}\left(D^2 Q(\mathbf{q}^\star)\right)^{-1}$ and $\mathbf{B}_s = \nabla g_s(\mathbf{q}^\star)$.*

**Proof**: Define $\boldsymbol{\xi}$ as the current approximation $\mathbf{q}$. Let $G_s(\boldsymbol{\xi})$ be $q_s$ that maximizes the objective function $Q(\mathbf{q})$ under the constraint $\mathbf{g}_s(\mathbf{q}) = \mathbf{g}_s(\boldsymbol{\xi}) = [\boldsymbol{\xi}_{\backslash s}]$.

Let $M_0(\mathbf{q}) = \mathbf{q}$ and

$$M_s(\mathbf{q}) = G_s(M_{s-1}(\mathbf{q})) \qquad \text{for all } 1 \le s \le S. \tag{23}$$

Then by construction of VB, we have $\mathbf{q}^{(t+s/S)} = M_s(\mathbf{q}^{(t)})$, $s = 1, \ldots, S$ and $DM(\mathbf{q}^\star) = DM_S(\mathbf{q}^\star)$. At the stationary points, $\mathbf{q}^\star = DM_s(\mathbf{q}^\star)$ for all $s$.

We differentiate both sides of equation (23) and evaluate them at $\mathbf{q} = \mathbf{q}^\star$:

$$DM_s(\mathbf{q}) = DM_{s-1}(\mathbf{q})DG_S(M_{s-1}(\mathbf{q}^\star)) = DM_{s-1}(\mathbf{q}^\star)DG_S(\mathbf{q}^\star) \tag{24}$$

It follows that $DM(\mathbf{q}^\star) = \prod_{s=1}^{S} DG_S(\mathbf{q}^\star)$.

To calculate $DG_S(\mathbf{q}^\star)$, we differentiate the constraint $\mathbf{g}_s(G_s(\boldsymbol{\xi})) = \mathbf{g}_s(\boldsymbol{\xi})$ and evaluate both sides at $\boldsymbol{\xi} = \mathbf{q}^\star$, such that

$$DG_s(\mathbf{q}^\star)\mathbf{B}_s = \mathbf{B}_s. \tag{25}$$

Similarly, we differentiate the Lagrange equation $DQ_s(G(\boldsymbol{\xi})) - \nabla g_s(G(\boldsymbol{\xi}))\lambda_s(\boldsymbol{\xi}) = 0$ and evaluate both sides at $\boldsymbol{\xi} = \mathbf{q}^\star$. This yields

$$DG_s(\mathbf{q}^\star)D^2 Q_s(\mathbf{q}^\star) - D\lambda_s(\mathbf{q}^\star)\mathbf{B}_s^{\mathrm{T}} = 0 \tag{26}$$

Equation (26) holds because $\frac{\partial^2 \mathbf{g}_s}{\partial q_i \partial q_j} = \mathbf{0}$.

Combining (25) and (26) yields

$$DG_s(\mathbf{q}^\star) = \mathbf{B}_s[\mathbf{B}_s^{\mathrm{T}}\left(D^2 Q_s(\mathbf{q}^\star)\right)^{-1}\mathbf{B}_s]^{-1}\mathbf{B}_s^{\mathrm{T}}\left(D^2 Q_s(\mathbf{q}^\star)\right)^{-1}.\square \tag{27}$$

In the $s$ update we fix $\mathbf{q}_{\backslash s}$, i.e., $g_s(\mathbf{q}) = \mathbf{q}_{\backslash s}$. Therefore, $\mathbf{B}_s$ is an identity matrix with its $s^{th}$ column removed $\mathbf{B}_s = \mathbf{I}_{:,\backslash s}$, where $\mathbf{I}$ is the identity matrix and $\mathbf{s}, :$ means without the $s^{th}$ column.

Denote $C_s = \left(D^2 Q_s(\mathbf{q}^\star)\right)^{-1}$. Without the loss of generality, we set $s = S$. It is easy to obtain

$$\mathbf{B}_S^{\mathrm{T}}C\mathbf{B}_S = C_{\backslash S, \backslash S} \tag{28}$$

where $\backslash S, \backslash S$ means without row $S$ and column $S$.

Inserting (28) into (27) yields

$$P_S = DG_S(\mathbf{q}^\star) = \begin{pmatrix} \mathbf{I}_{d-1} & C_{\backslash S, \backslash S}^{-1} C_{\backslash S, S} \\ \mathbf{0} & 0 \end{pmatrix} = \begin{pmatrix} \mathbf{I}_{d-1} & -D^2 Q_{\backslash S, S}(D^2 Q_{S,S})^{-1} \\ \mathbf{0} & 0 \end{pmatrix} \tag{29}$$

where $\mathbf{I}_{d-1}$ is a $(d-1)$ by $(d-1)$ identity matrix, and $D^2 Q_{\backslash S, S} = \frac{\partial^2 Q(q_{\mathbf{q}}(\mathbf{x}) \| p(\mathbf{x}))}{\partial \mathbf{q}_{\backslash S}^{\mathrm{T}} \partial \mathbf{q}_S}$ and $D^2 Q_{S,S} = \frac{\partial^2 Q(q_{\mathbf{q}}(\mathbf{x}) \| p(\mathbf{x}))}{\partial \mathbf{q}_S^{\mathrm{T}} \partial \mathbf{q}_S}$. Notice that we use Schur complements to obtain (29). Similar to the calculation of $P_S$ via (29), we can derive $P_s$ for $s = 1, \ldots, S-1$ with structures similar to $P_S$.

The above results help us understand the convergence speed of VB. For example, we have

$$\mathbf{q}^{(t+1)} - \mathbf{q}^\star = P_S^{\mathrm{T}} \cdots P_1^{\mathrm{T}}(\mathbf{q}^{(t)} - \mathbf{q}^\star). \qquad (30)$$

For $\mathbf{q}_S$, $\mathbf{q}_S^{(t+1)} - \mathbf{q}_S^\star = \big( -(D^2 Q_{S,S})^{-1} D^2 Q_{S,\backslash S} \quad 0 \big)(\mathbf{q}^{(t+(S-1)/S)} - \mathbf{q}^\star)$.

Clearly, if we view $D^2 Q_{S,\backslash S}$ as the correlation between $q_S$ and $q_{\backslash S}$, then the smaller "correlation", the faster the convergence. In the extreme case, if there is no correlation between $q_S$ and $q_{\backslash S}$, then $\mathbf{q}_S^{(t+1)} - \mathbf{q}_S^\star = 0$ after the first iteration. Since the global convergence rate is bounded by the maximal component convergence rate and generally there are many components with convergence rate same as the global rate. Therefore, the instant convergence of $q_S$ could help increase the global convergence rate.

For PX-VB, we can compute the matrix rate of convergence similarly. In the toy example in Section 2, PX-VB introduces an auxiliary variable $\alpha$ which has zero correlation with $w$, leading an instant convergence of the algorithm. This suggests that PX-VB improves the convergence by reducing the correlation among $\{q_s\}$. Rigorously speaking, the reduction step in PX-VB implictly defines a mapping between $\mathbf{q}$ to $q_{\alpha_0}$ through the auxiliary variables $\boldsymbol{\alpha}$: $(\mathbf{q}, p_{\alpha_0}) \rightarrow (\mathbf{q}, p_{\boldsymbol{\alpha}}) \rightarrow (\mathbf{q}_{\boldsymbol{\alpha}}, p_{\alpha_0})$. Denote this mapping as $M_{\boldsymbol{\alpha}}$ such as $\mathbf{q}_{\boldsymbol{\alpha}} = M_{\boldsymbol{\alpha}}(\mathbf{q})$. Then we have $DM_x(\mathbf{q}^\star) = DG_1(\mathbf{q}^\star) \cdots DG_{\boldsymbol{\alpha}}(\mathbf{q}^\star) \cdots DG_S(\mathbf{q}^\star)$

It is known that the spectral norm has the following submultiplicative property $\|EF\| <= \|E\|\|F\|$, where $E$ and $F$ are two matrices. Thus, as long as the largest eigenvalue of $M_{\boldsymbol{\alpha}}$ is smaller than 1, PX-VB converges faster than VB. The choice of $\boldsymbol{\alpha}$ affects the convergence rate by controlling the eigenvalue of this mapping. The smaller the largest eigenvalue of $M_{\boldsymbol{\alpha}}$, the faster PX-VB converges. In practice, we can check this eigenvalue to make sure the constructed PX-VB algorithm enjoys a fast convergence rate.

## 5   Discussion

We have provided a general approach to speeding up convergence of variational Bayesian learning. Faster convergence is guaranteed theoretically provided that the Jacobian of the transformation from auxiliary parameters to variational components has spectral norm bounded by one. This property can be verified in each case separately. Our empirical results show that the performance gain due to the auxiliary method is substantial.

**Acknowledgments**

T. S. Jaakkola was supported by DARPA Transfer Learning program.

## References

Beal, M. (2003). *Variational algorithms for approximate Bayesian inference*. Doctoral dissertation, Gatsby Computational Neuroscience Unit, University College London.

Bishop, C., & Tipping, M. E. (2000). Variational relevance vector machines. *16th UAI*.

Jordan, M. I., Ghahramani, Z., Jaakkola, T. S., & Saul, L. K. (1998). An introduction to variational methods in graphical models. *Learning in Graphical Models*. http://www.ai.mit.edu/~tommi/papers.html.

Liu, C., Rubin, D. B., & Wu, Y. N. (1998). Parameter expansion to accelerate EM: the PX-EM algorithm. *Biometrika*, *85*, 755–770.

Liu, J. S., & Wu, Y. N. (1999). Parameter expansion for data augmentation. *Journal of the American Statistical Association*, *94*, 1264–1274.

Luo, Z. Q., & Tseng, P. (1992). On the convergence of the coordinate descent method for convex differentiable minimization. *Journal of Optimization Theory and Applications*, *72*, 7–35.

MacKay, D. J. (1992). Bayesian interpolation. *Neural Computation*, *4*, 415–447.

Salakhutdinov, R., Roweis, S. T., & Ghahramani, Z. (2003). Optimization with EM and Expectation-Conjugate-Gradient. *Proceedings of International Conference on Machine Learning*.

Tipping, M. E. (2000). The relevance vector machine. *NIPS* (pp. 652–658). The MIT Press.

van Dyk, D. A., & Meng, X. L. (2001). The art of data augmentation (with discussion). *Journal of Computational and Graphical Statistics*, *10*, 1–111.
